# Synchrony Detection by Analogue VLSI Neurons with Bimodal STDP Synapses

**Adria Bofill-i-Petit**
The University of Edinburgh
Edinburgh, EH9 3JL
Scotland
adria.bofill@ee.ed.ac.uk

**Alan F. Murray**
The University of Edinburgh
Edinburgh, EH9 3JL
Scotland
alan.murray@ee.ed.ac.uk

## Abstract

We present test results from spike-timing correlation learning experiments carried out with silicon neurons with STDP (Spike Timing Dependent Plasticity) synapses. The weight change scheme of the STDP synapses can be set to either weight-independent or weight-dependent mode. We present results that characterise the learning window implemented for both modes of operation. When presented with spike trains with different types of synchronisation the neurons develop bimodal weight distributions. We also show that a 2-layered network of silicon spiking neurons with STDP synapses can perform hierarchical synchrony detection.

## 1  Introduction

Traditionally, Hebbian learning algorithms have interpreted Hebb's postulate in terms of coincidence detection. They are based on mean spike firing rates correlations between presynaptic and postsynaptic spikes rather than upon precise timing differences between presynaptic and postsynaptic spikes.

In recent years, new forms of synaptic plasticity that rely on precise spike-timing differences between presynaptic and postsynaptic spikes have been discovered in several biological systems[1][2][3]. These forms of plasticity, generally termed Spike Timing Dependent Plasticity (STDP), increase the synaptic efficacy of a synapse when a presynaptic spike reaches the neuron a few milliseconds before the postsynaptic action potential. In contrast, when the postsynaptic neuron fires immediately before the presynaptic neuron the strength of the synapse diminishes.

Much debate has taken place regarding the precise characteristics of the learning rules underlying STDP [4]. The presence of weight dependence in the learning rule has been identified as having a dramatic effect on the computational properties of STDP. When weight modifications are independent of the weight value, a strong competition takes places between the synapses. Hence, even when no spike-timing correlation is present in the input, synapses develop maximum or minimum strength so that a bimodal weight distribution emerges from learning[5]. Conversely, if the learning rule is strongly weight-dependent, such that strong synapses receive less potentiation than weaker ones while depression is independent of the synaptic strength,

a smooth unimodal weight distribution emerges from the learning process[6].

In this paper we present circuits to support STDP on silicon. Bimodal weight distributions are effectively binary. Hence, they are suited to analog VLSI implementation, as the main barrier to the implementation of on-chip learning, the long term storage of precise analog weight values, can be rendered unimportant. However, weight-independent STDP creates a highly unstable learning process that may hinder learning when only low levels of spike-timing correlations exist and neurons have few synapses. The circuits proposed here introduce a tunable weight dependence mechanism which stabilises the learning process. This allows finer correlations to be detected than does a weight-independent scheme. In the weight-dependent learning experiments reported here the weight-dependence is set at moderate levels such that bimodal weight distributions still result from learning.

The analogue VLSI implementation of spike-based learning was first investigated in [7]. The authors used a weight-dependent scheme and concentrated on the weight normalisation properties of the learning rule. In [8], we proposed circuits to implement asymmetric STDP which lacked the weight-dependent mechanism. More recently, others have also investigated asymmetric STDP learning using VLSI systems[9][10]. STDP synapses that contain an explicit bistable mechanism have been proposed in [10]. Long-term bistable synapses are a good technological solution for weight storage. However, the maximum and minimum weight limits in bimodal STDP already act as natural attractors. An explicit bistable mechanism may increase the instability of the learning process and may hinder, in consequence, the detection of subtle correlations. In contrast, the circuits that we propose here introduce a mechanism that tends to stabilise learning.

## 2  STDP circuits

The circuits in Figure 1 implement the asymmetric decaying learning window with the abrupt transition at the origin that is so characteristic of STDP. The weight of each synapse is represented by the charge stored on its weight capacitor $C_w$. The strength of the weight is inversely proportional to $V_w$. The closer the value of $V_w$ is to GND , the stronger is the synapse.

Our silicon spiking neurons signal their firing events with the sequence of pulses seen in Figure 1c. Signal *post_bp* is back-propagated to the afferent synapses of the neuron. *Long* is a longer pulse (a few $\mu$s) used in the current neuron (termed as signal *postLong* in Figure 1b). *Long* is also sent to input synapses of following neurons in the activity path (see *preLong* in 1a). Finally, *spikeOut* is the presynaptic spike for the next receiving neuron (termed *pre* in Figure 1a). More details on the implementation of the silicon neuron can be found in [11]

In Figure 1a, if *preLong* is long enough (a few $\mu$s) the voltage created by $I_{bpot}$ on the diode connected transistor N5 is copied to the gate of N2. This voltage across $C_{pot}$ decays with time from its peak value due to a leakage current set by $V_{bpot}$. When the postsynaptic neuron fires, a back propagation pulse *post_bp* switches N3 on. Therefore, the weight is potentiated ($V_w$ decreased) by an amount which reflects the time elapsed since the last presynaptic event.

A weight dependence mechanism is introduced by the simple linearised V-I configuration P5-P6 and current mirror N7-N6 (see Figure 1a). P5 is a low gain transistor operated in strong inversion whereas P6 is a wide transistor made to operate in weak inversion such that it has even higher gain. When the value of $V_w$ decreases (weight increase) the current through P5-P6 increases, but P5 is maintained in the linear region by the high gain transistor. Thus, a current proportional to the value of the weight is subtracted from $I_{bpot}$. The resulting smaller current injected into N5 will cause a drop in the peak of potentiation for large weight values.

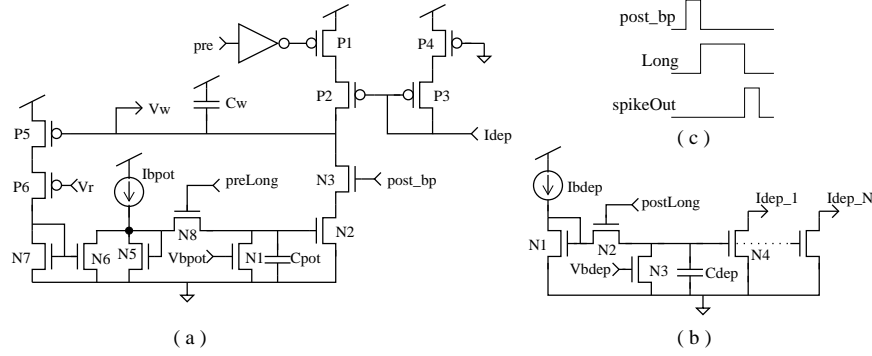

Figure 1: *Weight change circuits. (**a**) The strength of the synapse is inversely proportional to the value of $V_w$. The lower $V_w$, the smaller the weight of the synapse. This section of the weight change circuit detects* causal *spike correlations. (**b**) A single* depression *circuit present in the soma of the neuron creates the decaying shape of the depression side of the learning window. (**c**) Waveforms of pulses that signal an action potential event. They are used to stimulate the weight change circuits.*

In a similar manner to potentiation, the weight is weakened by the circuit of Figure 1b when it detects a *non-causal* interaction between a presynaptic and a postsynaptic spike. When a postsynaptic spike event is generated a postLong pulse charges $C_{dep}$. The charge accumulated leaks linearly through N3 at a rate set by $V_{bdep}$. A set of non-linear decaying currents ($I_{depX}$) is sent to the weight change circuits placed in the input synapse (see $I_{dep}$ in Figure 1a). When a presynaptic spike reaches a synapse P1 is switched on. If this occurs soon enough after the *postLong* pulse was generated, $V_w$ is brought closer to Vdd (weight strength decreased). Only one *depression* circuit per neuron is required since the depression part of the learning rule is independent of the weight value.

A chip including 5 spiking neurons with STDP synapses has been fabricated using a standard $0.6\mu m$ CMOS process. Each neuron has 6 learning synapses, a single excitatory non-learning synapse and a single inhibitory one. Along with the silicon neuron circuits, the chip contains several voltage buffers that allow us to monitor the behaviour of the neuron. The testing setup uses a networked logic analysis system to stimulate the silicon neuron and to capture the results of on-chip learning. An externally addressable circuit creates *preLong* and *pre* pulses to stimulate the synapses.

## 3   Weight-independent learning rule

### 3.1   Characterisation

A weight-independent weight change regime is obtained by setting $V_r$ to Vdd in the weight change circuit presented in Figure 1 . The resulting learning window on silicon can be seen in Figure 2. Each point in the curve was obtained from the stimulation of the fix synapse and a learning synapse with a varying delay between them. As can be seen in the figure, the circuit is highly tunable. Figure 2a shows that the peaks for potentiation and depression can be set independently. Also, as shown in Figure 2b the decay of the learning window for both sides of the curve can be set independently of the maximum weight change with $V_{bdep}$ and $V_{bpot}$. Since the weight-dependent mechanism is switched off, the curve of the learning window is the same for a wide range of $V_w$. Obviously, when the weight voltage $V_w$ approaches

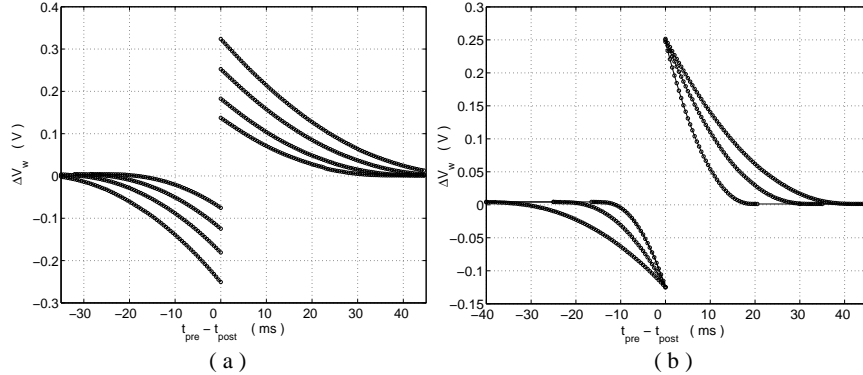

Figure 2: *Experimental learning window for weight-independent STDP. The curves show the weight modification induced in the weight of a learning synapse for different time intervals between the presynaptic and the postsynaptic spike. For the results shown, the synapses were operated in a weight-independent mode. (a) The peaks of the learning window is shown for 4 different settings. The peak for potentiation and depression are tuned independently with $I_{bpot}$ and $I_{bdep}$ (b) The rate of decay of the learning window for potentiation and depression can be set independently without affecting the maximum weight change.*

any of the power supply rails a saturation effect occurs as the transistors injecting current in the weight capacitor leave saturation. For the learning experiment with weight-independent weight change the area under the potentiation curve should be approximately 50% smaller than the area under the depression region.

## 3.2 Learning spike-timing correlations with weight-independent learning

We stimulated a 6-synapse silicon neuron with 6 independent Poisson-distributed spike trains with a rate of 30Hz. An absolute refractory period of 10ms was enforced between consecutive spikes of each train. Refractoriness helps break the temporal axis into disjoint segments so that presynaptic spikes can make less noisy "predictions" of the postsynaptic time of firing. We introduced spike-timing correlations between the inputs for synapses 1 and 2. Synapses 3 to 6 were uncorrelated.

The evolution of the 6 weights for one of such experiments is show in Figure 3. The correlated inputs shared 35% of the spike-timings. They were constructed by merging two independent 19.5Hz Poisson-distributed spike trains with a common 10.5Hz spike train. As can be seen in Figure 3 the weights of synapses that receive correlated activity reach maximum strength ($V_w$ close to GND) whereas the rest decay towards Vdd. Clearly, the bimodal weight distribution reflects the correlation pattern of the input signals.

## 3.3 Hierarchical synchrony detection

To experiment with hierarchical synchrony detection we included in the chip a small 2-layered network of STDP silicon neurons with the configuration shown in Figure 4. Neurons in the first layer were stimulated with independent sets of Poisson-distributed spike trains with a mean spiking rate of 30Hz. As with the experiments presented in the preceding section, a 10ms refractory period was forced between consecutive spikes. A primary level of correlation was introduced for each neuron in the first layer as signalled by the arrowed bridge between the

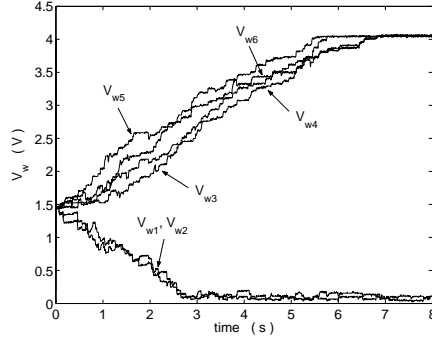

Figure 3: *Learning experiment with weight-independent STDP.*

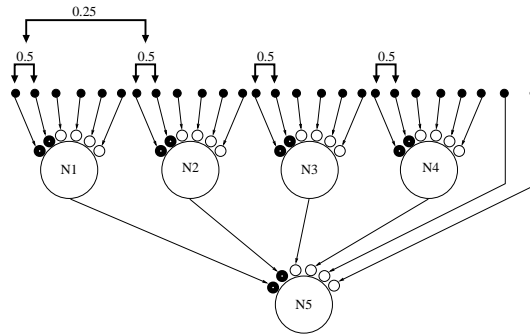

Figure 4: *Final weight values for a 2-layered network of STDP silicon neurons.*

inputs of synapses 1 and 2 of each neuron. For the results shown here these 2 inputs of each neuron shared 50% of the spike-timings (indicated with 0.5 on top of the double-arrowed bridge of Figure 4). A secondary level of correlation was introduced between the inputs of synapses 1 and 2 of both N1 and N2, as signalled by the arrow linking the first level of correlations of N1 and N2. This second level of correlations is weaker, with only 25% of shared spikes (indicated with 0.25 in Figure 4). The two direct inputs of N5, in the second layer, were also Poisson distributed but had a rate of 15Hz.

The evolution of the weights recorded for the experiment just described is presented in Figure 5. On the left, we see the weight evolution for N1. The weights corresponding to synapses 1 and 2 evolve towards the maximum value (i.e. GND). The weights of the remaining synapses, which receive random activity, decrease (i.e. $V_w$ close to Vdd). The other neurons in the 1st layer have weight evolutions similar to that of N1. Synapses with synchronised activity corresponding to the 1st level of correlations win the competition imposed by STDP. The $V_w$ traces on the right-hand side of Figure 5 show how N5 in the second layer captures the secondary level of correlation. Weights of the synapses receiving input from N1 and N2 are reinforced while the rest are decreased towards the minimum possible weight value ($V_w$ = Vdd). Clearly, the second layer only captures features from signals which have already a basic level of interesting features (primary level of correlations) detected by the first layer.

In Figure 4, we have represented graphically the final weight distribution for all synapses. As marked by filled circles, only synapses in the path of hierarchical

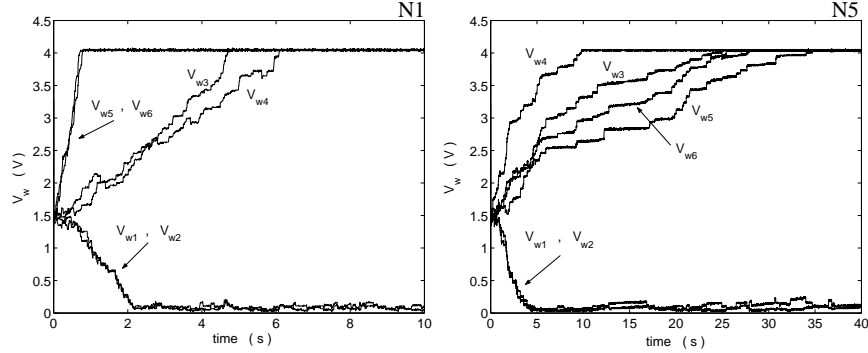

Figure 5: *Hierarchical synchrony detection. (a) Weight evolution of neuron in first layer. (b) Weight evolution of output neuron in 2nd layer.*

synchrony activity develop maximum weight strength. In contrast, weights with final minimum strength are indicated by empty circles. These correspond to synapses of first layer neurons which received uncorrelated inputs or synapses of N5 which received inputs from neurons stimulated without a secondary level of correlations (N3-N4).

# 4   Weight-dependent learning rule

## 4.1   Characterisation

The STDP synapses presented can also be operated in weight-dependent mode. The weight dependent learning window implemented is similar to that which seems to underly some STDP recordings from biological neurons [6]. Figure 6a shows chip results of the weight-dependent learning rule. The weight change curve for potentiation is given for 3 different weight values. The larger the weight value (low $V_w$), the smaller the degree of potentiation induced in the synapse. The depression side of the learning window is unaffected by the weight value since the depression circuit shown in Figure 1b does not have an explicit weight-dependent mechanism.

## 4.2   Learning spike-timing correlations with weight-dependent learning

Figure 6b shows the weight evolution for an experiment where the correlated activity between synapses 1 and 2 consisted of only 20% of common spike-timings. As in the weight-independent experiments, the mean firing rate was 30Hz and a refractory period of 10ms was enforced.

Finally, we stimulated a neuron in weight-dependent mode with a form of synchrony where spike-timings coincided in a time window (*window of correlation*) instead of being perfectly matched (syn0-1). The uncorrelated inputs (syn2-5) were Poisson-distributed spike trains. The synchrony data was an inhomogeneous Poisson spike train with a rate modulated by a binary signal with random transition points. Figure 7 shows a normalised histogram of spike intervals between the correlated inputs for synapses 0 and 1 (Figure 7a) and the histogram of the uncorrelated inputs for synapses 2 and 3 (Figure 7b). Again, as can be seen in Figure 7c the neuron with weight-dependent STDP can detect this low-level of synchrony with non-coincident spikes. Clearly, the bimodal weight distribution identifies the syn-

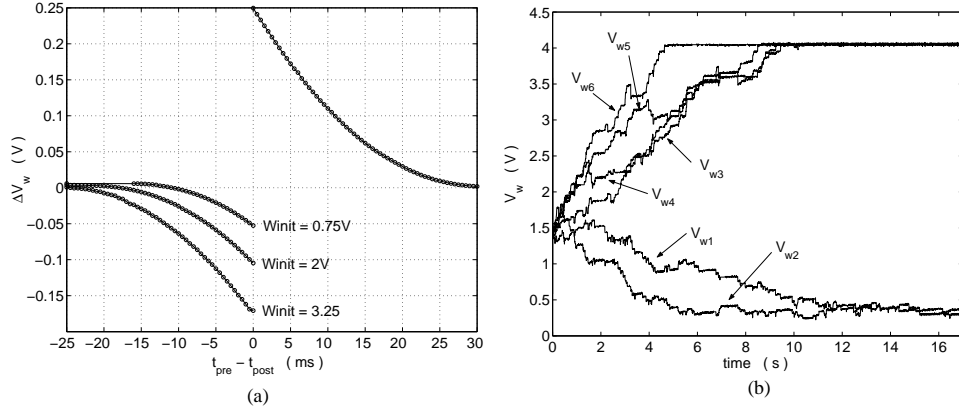

Figure 6: *(a) Experimental learning window for weight-dependent STDP (b) Learning experiment with weight-dependent STDP. Synapses 1 and 2 share 20% of spike-timings. The other synapses receive completely uncorrelated activity. Correlated activity causes synapses to develop strong weights ($V_w$ close to GND).*

chrony pattern of the inputs.

## 5    Conclusions

The circuits presented can be used to study both weight-dependent and weight-independent learning rules. The influence of weight-dependence on the final weight distribution has been studied extensively[5][6]. In this paper, we have concentrated on the stabilising effect that moderate weight-dependence can have on learning processes that develop bimodal weight distributions. By introducing weight-dependence subtle spike-timing correlations can be detected.

We have also shown experimentally that a small feed-forward network of silicon neurons with STDP synapses can detect a hierarchical synchrony structure embedded in noisy spike trains.

We are currently investigating the synchrony amplification properties of silicon neurons with bimodal STDP. We are also working on a new chip that uses lateral-inhibitory connections between neurons to classify data with complex synchrony patterns.

## References

[1] G-Q. Bi and M m Poo. Synaptic modifications in cultured hippocampal neurons; dependence on spike timing, synaptic strength and postsynaptic cell type. *Journal of Neuroscience*, 18:10464–10472, 1998.

[2] L.I. Zhang, H.W. Tao, C.E. Holt, W.A. Harris, and M m. Poo. A critical window for cooperation and competition among developing retinotectal synapses. *Nature*, 395:37–44, 1998.

[3] H. Markram, J. Lubke, M. Frotscher, and B. Sakmann. Regulation of synaptic efficacy by coincidence of postsynaptic APs and EPSPs. *Science*, 275:213–215, 1997.

[4] A. Kepecs, M.C.W van Rossum, S. Song, and J. Tegner. Spike-timing-dependent plasticity: common themes and divergent vistas. *Biological Cybernetics*, 87:446–458, 2002.

[5] S. Song, K.D. Miller, and L.F. Abbott. Competitive Hebbian learning through spike-timing dependent synaptic plasticity. *Nature Neuroscience*, 3:919–926, 2000.

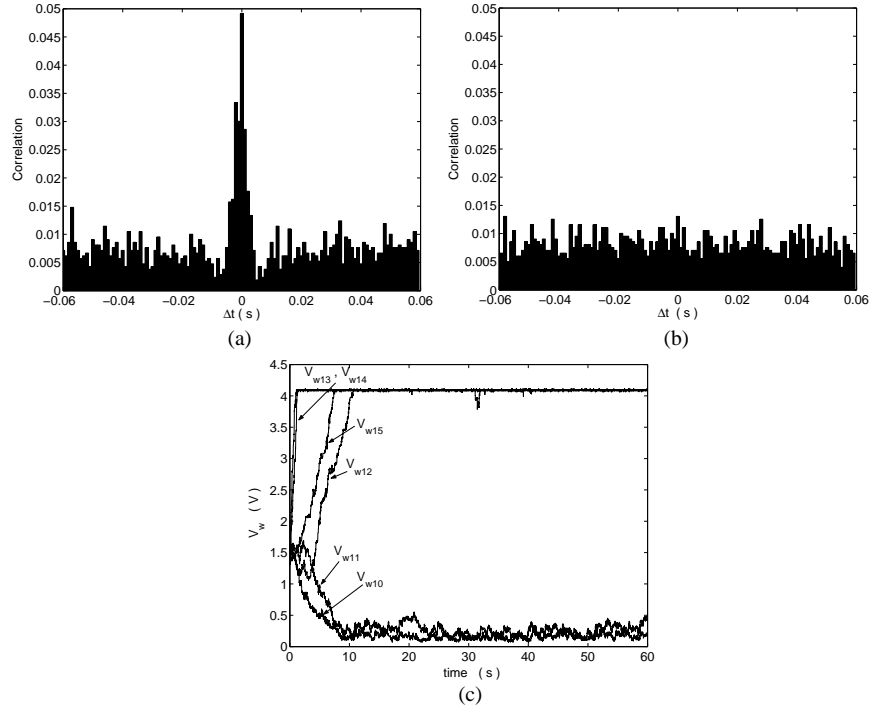

Figure 7:  *Detection of non-coincident spike-timing synchrony with weight-dependent STDP.* *(a)* *Normalised spike interval histogram of the 2 correlated inputs (synapses 0 and 1).* *(b)* *Normalised spike interval histogram between 2 uncorrelated inputs (synapses 2-5)* *(c)* *Synapses 0 and 1 win the learning competition.*

[6] M. van Rossum and G.G. Turrigiano. Corrrelation based learning from spike timing dependent plasticity. *Neurocomputing*, 38-40:409–415, 2001.

[7] P. Halfiger, M. Mahowald, and L. Watts. A spike based learning neuron in analog VLSI. In M.C. Mozer, M.I. Jordan, and T. Petsche, editors, *Advances in Neural Information Processing Systems 9*, pages 692–698. MIT Press, 1996.

[8] A. Bofill, A. F. Murray, and D. P. Thompson. Circuits for VLSI implementation of temporally asymmetric Hebbian learning. In T. G. Dietterich, S. Becker, and Z. Ghahramani, editors, *Advances in Neural Information Processing Systems 14*. MIT Press, 2002.

[9] R. J. Vogelstein, F. Tenore, R. Philipp, M. S. Adlerstein, D. H. Goldberg, and G. Cauwenberghs. Spike timing-dependent plasticity in the address domain. In S. Becker, S. Thrun, and Klaus Obermayer, editors, *Advances in Neural Information Processing Systems 15*. MIT Press, 2003.

[10] G. Indiveri. Circuits for bistable spike-timing-dependent plasticity neuromorphic vlsi synapses. In S. Becker, S. Thrun, and Klaus Obermayer, editors, *Advances in Neural Information Processing Systems 15*. MIT Press, 2003.

[11] A. Bofill i Petit and A.F. Murray. Learning temporal correlations in biologically-inspired aVLSI. In *IEEE Internation Symposium on Circuits and Systems*, volume 5, pages 817–820, 2003.
